# Local Phase Coherence
# and the Perception of Blur

**Zhou Wang** and **Eero P. Simoncelli**
Howard Hughes Medical Institute
Center for Neural Science and Courant Institute of Mathematical Sciences
New York University, New York, NY 10003
zhouwang@ieee.org, eero.simoncelli@nyu.edu

Humans are able to detect blurring of visual images, but the mechanism by which they do so is not clear. A traditional view is that a blurred image looks "unnatural" because of the reduction in energy (either globally or locally) at high frequencies. In this paper, we propose that the disruption of local phase can provide an alternative explanation for blur perception. We show that precisely localized features such as step edges result in strong local phase coherence structures across scale and space in the complex wavelet transform domain, and blurring causes loss of such phase coherence. We propose a technique for coarse-to-fine phase prediction of wavelet coefficients, and observe that (1) such predictions are highly effective in natural images, (2) phase coherence increases with the strength of image features, and (3) blurring disrupts the phase coherence relationship in images. We thus lay the groundwork for a new theory of perceptual blur estimation, as well as a variety of algorithms for restoration and manipulation of photographic images.

## 1 Introduction

Blur is one of the most common forms of image distortion. It can arise from a variety of sources, such as atmospheric scatter, lens defocus, optical aberrations of the lens, and spatial and temporal sensor integration. Human observers are bothered by blur, and our visual systems are quite good at reporting whether an image appears blurred (or sharpened) [1, 2]. However, the mechanism by which this is accomplished is not well understood.

Clearly, detection of blur requires some model of what constitutes an unblurred image. In recent years, there has been a surge of interest in the modelling of natural images, both for purposes of improving the performance of image processing and computer vision systems, and also for furthering our understanding of biological visual systems. Early statistical models were almost exclusively based on a description of global Fourier power spectra. Specifically, image spectra are found to follow a power law [3–5]. This model leads to an obvious method of detecting and compensating for blur. Specifically, blurring usually reduces the energy of high frequency components, and thus the power spectrum of a blurry image should fall faster than a typical natural image. The standard formulation of the "deblurring" problem, due to Wiener [6], aims to restore those high frequency components to their original amplitude. But this proposal is problematic, since individual images show significant variability in their Fourier amplitudes, both in their shape and in the rate at which

they fall [1]. In particular, simply reducing the number of sharp features (e.g., edges) in an image can lead to a steeper falloff in global amplitude spectrum, even though the image will still appear sharp [7]. Nevertheless, the visual system seems to be able to compensate for this when estimating blur [1, 2, 7].

Over the past two decades, researchers from many communities have converged on a view that images are better represented using bases of multi-scale bandpass oriented filters. These representations, loosely referred to as "wavelets", are effective at decoupling the high-order statistical features of natural images. In addition, they provide the most basic model for neurons in the primary visual cortex of mammals, which are presumably adapted to efficiently represent the visually relevant features of images. Many recent statistical image models in the wavelet domain are based on the amplitudes of the coefficients, and the relationship between the amplitudes of coefficients in local neighborhoods or across different scales [e.g. 8]. In both human and computer vision, the amplitudes of complex wavelets have been widely used as a mechanism for localizing/representing features [e.g. 9–13]. It has also been shown that the relative wavelet amplitude as a function of scale can be used to explain a number of subjective experiments on the perception of blur [7].

In this paper, we propose the disruption of local phase as an alternative and effective measure for the detection of blur. This seems counterintuitive, because when an image is blurred through convolution with a symmetric linear filter, the phase information in the (global) Fourier transform domain does not change at all. But we show that this is not true for *local* phase information.

In previous work, Fourier phase has been found to carry important information about image structures and features [14] and higher-order Fourier statistics have been used to examine the phase structure in natural images [15]. It has been pointed out that at the points of isolated even and odd symmetric features such as lines and step edges, the *arrival phases* of all Fourier harmonics are identical [11, 16]. *Phase congruency* [11, 17] provides a quantitative measure for the agreement of such phase alignment pattern. It has also been shown that maximum phase congruency feature detection is equivalent to maximum local energy model [18]. Local phase has been used in a number of machine vision and image processing applications, such as estimation of image motion [19] and disparity [20], description of image textures [21], and recognition of persons using iris patterns [22]. However, the behaviors of local phase at different scales in the vicinity of image features, and the means by which blur affects such behaviors have not been deeply investigated.

## 2 Local Phase Coherence of Isolated Features

Wavelet transforms provide a convenient framework for localized representation of signals simultaneously in space and frequency. The wavelets are dilated/contracted and translated versions of a "mother wavelet" $w(x)$. In this paper, we consider symmetric (linear phase) wavelets whose mother wavelets may be written as a modulation of a low-pass filter:

$$w(x) = g(x)\, e^{j\omega_c x}\,, \tag{1}$$

where $\omega_c$ is the center frequency of the modulated band-pass filter, and $g(x)$ is a slowly varying and symmetric function. The family of wavelets derived from the mother wavelet are then

$$w_{s,p}(x) = \frac{1}{\sqrt{s}}\, w\left(\frac{x-p}{s}\right) = \frac{1}{\sqrt{s}}\, g\left(\frac{x-p}{s}\right)\, e^{j\omega_c(x-p)/s}\,, \tag{2}$$

where $s \in R^+$ is the scale factor, and $p \in R$ is the translation factor. Considering the fact that $g(-x) = g(x)$, the wavelet transform of a given real signal $f(x)$ can be written as

$$F(s,p) = \int_{-\infty}^{\infty} f(x)\, w_{s,p}^*(x)\, dx = \left[f(x) * \frac{1}{\sqrt{s}}\, g\left(\frac{x}{s}\right)\, e^{j\omega_c x/s}\right]_{x=p}\,. \tag{3}$$

Now assume that the signal $f(x)$ being analyzed is localized near the position $x_0$, and we rewrite it into a function $f_0(x)$ that satisfies $f(x) = f_0(x - x_0)$. Using the convolution theorem and the shifting and scaling properties of the Fourier transform, we can write

$$\begin{aligned} F(s,p) &= \frac{1}{2\pi} \int_{-\infty}^{\infty} F(\omega)\sqrt{s}\, G(s\,\omega - \omega_c)\, e^{j\omega p}\, d\omega \\ &= \frac{1}{2\pi} \int_{-\infty}^{\infty} F_0(\omega)\sqrt{s}\, G(s\,\omega - \omega_c)\, e^{j\omega(p-x_0)}\, d\omega \\ &= \frac{1}{2\pi\sqrt{s}} \int_{-\infty}^{\infty} F_0\left(\frac{\omega}{s}\right) G(\omega - \omega_c)\, e^{j\omega(p-x_0)/s}\, d\omega \,, \end{aligned} \tag{4}$$

where $F(\omega)$, $F_0(\omega)$ and $G(\omega)$ are the Fourier transforms of $f(x)$, $f_0(x)$ and $g(x)$, respectively.

We now examine how the phase of $F(s,p)$ evolves across space $p$ and scale $s$. From Eq. (4), we see that the phase of $F(s,p)$ highly depends on the nature of $F_0(\omega)$. If $F_0(\omega)$ is scale-invariant, meaning that

$$F_0\left(\frac{\omega}{s}\right) = K(s)F_0(\omega)\,, \tag{5}$$

where $K(s)$ is a real function of only $s$, but independent of $\omega$, then from Eq. (4) and Eq. (5) we obtain

$$\begin{aligned} F(s,p) &= \frac{K(s)}{2\pi\sqrt{s}} \int_{-\infty}^{\infty} F_0(\omega)\, G(\omega - \omega_c)\, e^{j\omega(p-x_0)/s}\, d\omega \\ &= \frac{K(s)}{\sqrt{s}} F(1, x_0 + \frac{p - x_0}{s})\,. \end{aligned} \tag{6}$$

Since both $K(s)$ and $s$ are real, we can write the phase as:

$$\Phi(F(s,p)) = \Phi(F(1, x_0 + \frac{p - x_0}{s}))\,. \tag{7}$$

This equation suggests a strong phase coherence relationship across scale and space. An illustration is shown in Fig. 1(a), where it can be seen that equal-phase contours in the $(s,p)$ plane form straight lines defined by

$$x_0 + \frac{p - x_0}{s} = C\,, \tag{8}$$

where $C$ can be any real constant. Further, all these straight lines converge exactly at the location of the feature $x_0$. More generally, the phase at any given scale may be computed from the phase at any other scale by simply rescaling the position axis.

This phase coherence relationship relies on the scale-invariance property of Eq. (5) of the signal. Analytically, the only type of continuous spectrum signal that satisfies Eq. (5) follows a power law:

$$F_0(\omega) = K_0\,\omega^P\,. \tag{9}$$

In the spatial domain, the functions $f_0(x)$ that satisfy this scale-invariance condition include the step function $f_0(x) = K(u(x) - \frac{1}{2})$ (where $K$ is a constant and $F_0(\omega) = K/j\omega$) and its derivatives, such as the delta function $f_0(x) = K\delta(x)$ (where $K$ is a constant and $F_0(\omega) = K$). Notice that both functions of $f_0(x)$ are precisely localized in space.

Figure 1(b) shows that this precisely convergent phase behavior is disrupted by blurring. Specifically, if we convolve a sharp feature (e.g., an step edge) with a low-pass filter, the resulting signal will no longer satisfy the scale-invariant property of Eq. (5) and the phase coherence relationship of Eq. (7). Thus, a measure of phase coherence can be used to detect blur. Note that the phase congruency relationship [11, 17], which expresses the alignment of phase at the location of a feature, corresponds to the center (vertical) contour of Fig. 1, which remains intact after blurring. Thus, phase congruency measures [11, 17] provide no information about blur.

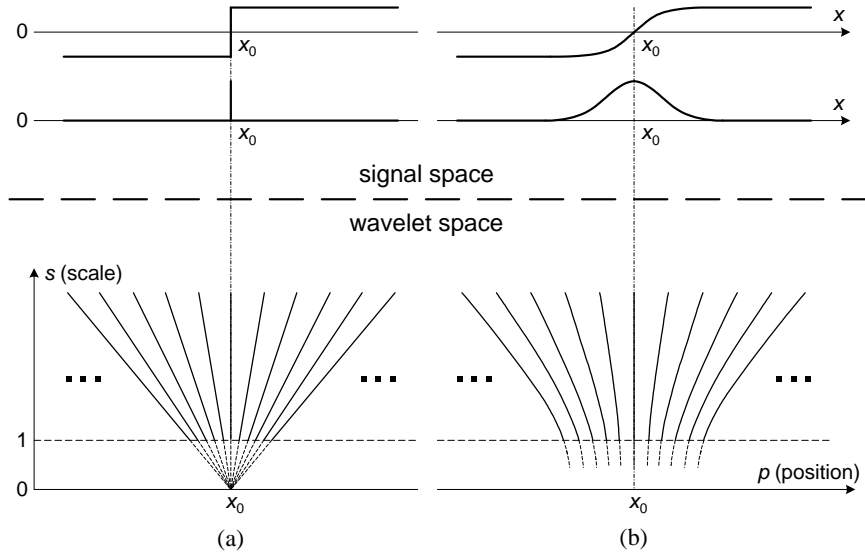

Fig. 1: Local phase coherence of precisely localized (scale-invariant) features, and the disruption of this coherence in the presence of blur. (a) precisely localized features. (b) blurred features.

## 3   Phase Prediction in Natural Images

In this section, we show that if the local image features are precisely localized (such as the delta and the step functions), then in the discrete wavelet transform domain, the phase of nearby fine-scale coefficients can be well predicted from their coarser-scale parent coefficients. We then examine these phase predictions in both sharp and blurred natural images.

### 3.1   Coarse-to-fine Phase Prediction

From Eq. (3), it is straightforward to prove that for $f_0(x) = K\delta(x)$,

$$\Phi(F(1,p)) = -\omega_c (p - x_0) + n_1\pi \,, \tag{10}$$

where $n_1$ is an integer whose value depends on the value range of $\omega_c (p - x_0)$ and the sign of $Kg(p - x_0)$. Using the phase coherence relation of Eq. (7), we have

$$\Phi(F(s,p)) = -\frac{\omega_c (p - x_0)}{s} + n_1\pi \,. \tag{11}$$

It can also be shown that for a step function $f_0(x) = K[u(x) - \frac{1}{2}]$, when $g(x)$ is slowly varying and $p$ is located near the feature location $x_0$,

$$\Phi(F(s,p)) \approx \frac{\omega_c (p - x_0)}{s} - \frac{\pi}{2} + n_2\pi \,. \tag{12}$$

Similarly, $n_2$ is an integer.

The discrete wavelet transform corresponds to a discrete sampling of the continuous wavelet transform $F(s,p)$. A typical sampling grid is illustrated in Fig. 2(a), where between every two adjacent scales, the scale factor $s$ doubles and the spatial sampling rate is halved. Now we consider three consecutive scales and group the neighboring coefficients $\{a, b_1, b_2, c_1, c_2, c_3, c_4\}$ as shown in Fig. 2(a), then it can be shown that the phases

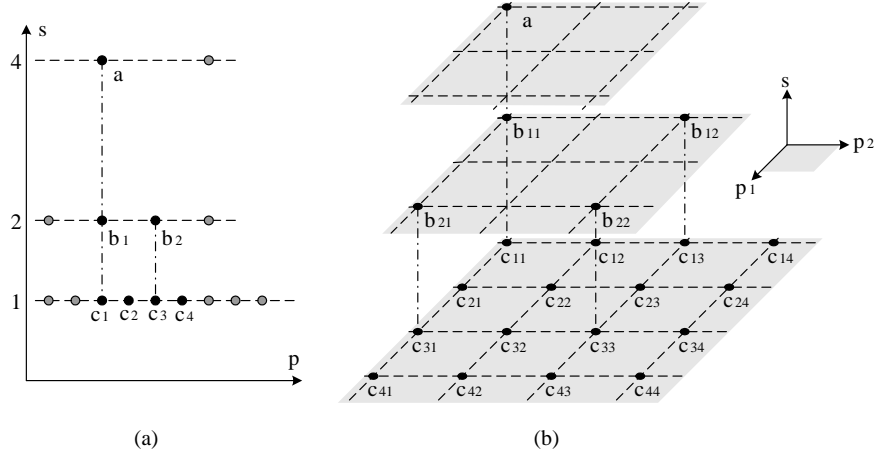

(a)                                    (b)

Fig. 2: Discrete wavelet transform sampling grid in the continuous wavelet transform domain. (a) 1-D sampling; (b) 2-D sampling.

of the finest scale coefficients $\{c_1, c_2, c_3, c_4\}$ can be well predicted from the coarser scale coefficients $\{a, b_1, b_2\}$, provided the local phase satisfies the phase coherence relationship. Specifically, the estimated phase $\hat{\Phi}$ for $\{c_1, c_2, c_3, c_4\}$ can be expressed as

$$\hat{\Phi}\begin{pmatrix} c_1 \\ c_2 \\ c_3 \\ c_4 \end{pmatrix} = \Phi\left( (a^*)^2 \cdot \begin{bmatrix} b_1^3 \\ b_1^2 b_2 \\ b_1 b_2^2 \\ b_2^3 \end{bmatrix} \right). \tag{13}$$

We can develop a similar technique for the two dimensional case. As shown in Fig. 2(b), the phase prediction expression from the coarser scale coefficients $\{a, b_{11}, b_{12}, b_{21}, b_{22}\}$ to the group of finest scale coefficients $\{c_{ij}\}$ is as follows:

$$\hat{\Phi}(\{c_{ij}\}) = \Phi\left( (a^*)^2 \cdot \begin{bmatrix} b_{11}^3 & b_{11}^2 b_{12} & b_{11} b_{12}^2 & b_{12}^3 \\ b_{11}^2 b_{21} & b_{11}^2 b_{22} & b_{11} b_{12} b_{22} & b_{12}^2 b_{22} \\ b_{11} b_{21}^2 & b_{11} b_{21} b_{22} & b_{11} b_{22}^2 & b_{12} b_{22}^2 \\ b_{21}^3 & b_{21}^2 b_{22} & b_{21} b_{22}^2 & b_{22}^3 \end{bmatrix} \right). \tag{14}$$

### 3.2   Image Statistics

We decompose the images using the "steerable pyramid" [23], a multi-scale wavelet decomposition whose basis functions are spatially localized, oriented, and roughly one octave in bandwidth. A 3-scale 8-orientation pyramid is calculated for each image, resulting in 26 subbands (24 oriented, plus highpass and lowpass residuals). Using Eq. (14), the phase of each coefficient in the 8 oriented finest-scale subbands is predicted from the phases of its coarser-scale parent and grandparent coefficients as illustrated in Fig. 2(b). We applied such a phase prediction method to a dataset of 1000 high-resolution sharp images as well as their blurred versions, and then examined the errors between the predicted and true phases at the fine scale.

The summary histograms are shown in Fig. 3. In order to demonstrate how blurring affects the phase prediction accuracy, in all these conditional histograms, the magnitude axis corresponds to the coefficient magnitudes of the original image, so that the same column in the three histograms correspond to the same set of coefficients in spatial location. From Fig.

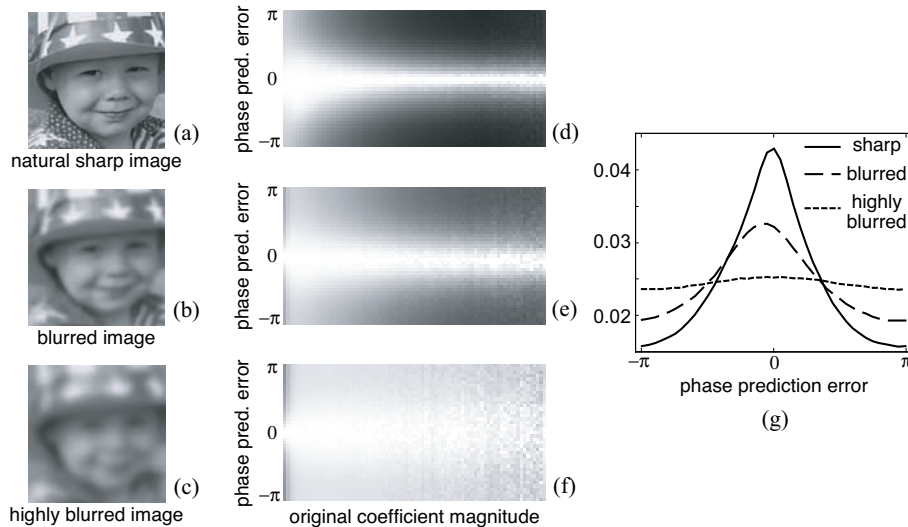

natural sharp image

blurred image

highly blurred image

original coefficient magnitude

phase pred. error

phase prediction error

Fig. 3: Local phase coherence statistics in sharp and blurred images. (a),(b),(c): example natural, blurred and highly blurred images taken from the test image database of 1000 (512×512, 8bits/pixel, gray-scale) natural images with a wide variety of contents (humans, animals, plants, landscapes, man-made objects, etc.). Images are cropped to 200×200 for visibility; (d),(e),(f): conditional histograms of phase prediction error as a function of the original coefficient magnitude for the three types of images. Each column of the histograms is scaled individually, such that the largest value of each column is mapped to white; (g) phase prediction error histogram of significant coefficients (magnitude greater than 20).

3, we observe that phase coherence is highly effective in natural images and the phase prediction error decreases as the coefficient magnitude increases. Larger coefficients implies stronger local phase coherence. Furthermore, as expected, the blurring process clearly reduces the phase prediction accuracy. We thus hypothesize that it is perhaps this disruption of local phase coherence that the visual system senses as being "unnatural".

## 4 Discussion

This paper proposes a new view of image blur based on the observation that blur induces distortion of local phase, in addition to the widely noted loss of high-frequency energy. We have shown that isolated precisely localized features create strong local phase coherence, and that blurring disrupts this phase coherence. We have also developed a particular measure of phase coherence based on coarse-to-fine phase prediction, and shown that this measure can serve as an indication of blur in natural images. In the future, it remains to be seen whether the visual systems detect blur by comparing the relative amplitude of localized filters at different scales [7], or alternatively, comparing the relative spread of local phase across scale and space.

The coarse-to-fine phase prediction method was developed in order to facilitate examination of phase coherence in real images, but the computations involved bear some resemblance to the behaviors of neurons in the primary visual cortex (area V1) of mammals. First, phase information is measured using pairs of localized bandpass filters in quadrature, as are widely used to describe the receptive field properties of neurons in mammalian primary visual cortex (area V1) [24]. Second, the responses of these filters must be ex-

ponentiated for comparison across different scales. Many recent models of V1 response incorporate such exponentiation [25]. Finally, responses are seen to be normalized by the magnitudes of neighboring filter responses. Similar "divisive normalization" mechanisms have been successfully used to account for many nonlinear behaviors of neurons in both visual and auditory neurons [26, 27]. Thus, it seems that mammalian visual systems are equipped with the basic computational building blocks that can be used to process local phase coherence.

The importance of local phase coherence in blur perception seems intuitively sensible from the perspective of visual function. In particular, the accurate localization of image features is critical to a variety of visual capabilities, including various forms of hyperacuity, stereopsis, and motion estimation. Since the localization of image features depends critically on phase coherence, and blurring disrupts phase coherence, blur would seem to be a particularly disturbing artifact. This perhaps explains the subjective feeling of frustration when confronted with a blurred image that cannot be corrected by visual accommodation.

For purposes of machine vision and image processing applications, we view the results of this paper as an important step towards the incorporation of phase properties into statistical models for images. We believe this is likely to lead to substantial improvements in a variety of applications, such as deblurring or sharpening by phase restoration, denoising by phase restoration, image compression, image quality assessment, and a variety of more creative photographic applications, such as image blending or compositing, reduction of dynamic range, or post-exposure adjustments of depth-of-field.

Furthermore, if we would like to detect the position of an isolated precisely localized feature from phase samples measured above a certain allowable scale, then infinite precision can be achieved using the phase convergence property illustrated in Fig. 1(a), provided the phase measurement is perfect. In other words, the detection precision is limited by the accuracy of phase measurement, rather than the highest spatial sampling density. This provides a workable mechanism of "seeing beyond the Nyquist limit" [28], which could explain a number of visual hyperacuity phenomena [29, 30], and may be used for the design of super-precision signal detection devices.

## References

[1] Y. Tadmor and D. J. Tolhurst, "Discrimination of changes in the second-order statistics of natural and synthetic images," *Vis Res*, vol. 34, no. 4, pp. 541–554, 1994.

[2] M. A. Webster, M. A. Georgeson, and S. M. Webster, "Neural adjustments to image blur," *Nature Neuroscience*, vol. 5, no. 9, pp. 839–840, 2002.

[3] E. R. Kretzmer, "The statistics of television signals," *Bell System Tech. J.*, vol. 31, pp. 751–763, 1952.

[4] D. J. Field, "Relations between the statistics of natural images and the response properties of cortical cells," *J. Opt. Soc. America*, vol. 4, pp. 2379–2394, 1987.

[5] D. L. Ruderman, "The statistics of natural images," *Network: Computation in Neural Systems*, vol. 5, pp. 517–548, 1996.

[6] N. Wiener, *Nonlinear Problems in Random Theory*. New York: John Wiley and Sons, 1958.

[7] D. J. Field and N. Brady, "Visual sensitivity, blur and the sources of variability in the amplitude spectra of natural scenes," *Vis Res*, vol. 37, no. 23, pp. 3367–3383, 1997.

[8] E. P. Simoncelli, "Statistical models for images: Compression, restoration and synthesis," in *Proc 31st Asilomar Conf on Signals, Systems and Computers*, (Pacific Grove, CA), pp. 673–678, Nov 1997.

[9] E. H. Adelson and J. R. Bergen, "Spatiotemporal energy models for the perception of motion," *J Optical Society*, vol. 2, pp. 284–299, Feb 1985.

[10] J. R. Bergen and E. H. Adelson, "Early vision and texture perception," *Nature*, vol. 333, pp. 363–364, 1988.

[11] M. C. Morrone and R. A. Owens, "Feature detection from local energy," *Pattern Recognition Letters*, vol. 6, pp. 303–313, 1987.

[12] N. Graham, *Visual pattern analyzers*. New York: Oxford University Press, 1989.

[13] P. Perona and J. Malik, "Detecting and localizing edges composed of steps, peaks and roofs," in *Proc. 3rd Int'l Conf Comp Vision*, (Osaka), pp. 52–57, 1990.

[14] A. V. Oppenheim and J. S. Lim, "The importance of phase in signals," *Proc. of the IEEE*, vol. 69, pp. 529–541, 1981.

[15] M. G. A. Thomson, "Visual coding and the phase structure of natural scenes," *Network: Comput. Neural Syst.*, no. 10, pp. 123–132, 1999.

[16] M. C. Morrone and D. C. Burr, "Feature detection in human vision: A phase-dependent energy model," *Proc. R. Soc. Lond. B*, vol. 235, pp. 221–245, 1988.

[17] P. Kovesi, "Phase congruency: A low-level image invariant," *Psych. Research*, vol. 64, pp. 136–148, 2000.

[18] S. Venkatesh and R. A. Owens, "An energy feature detection scheme," *Int'l Conf on Image Processing*, pp. 553–557, 1989.

[19] D. J. Fleet and A. D. Jepson, "Computation of component image velocity from local phase information," *Int'l J Computer Vision*, no. 5, pp. 77–104, 1990.

[20] D. J. Fleet, "Phase-based disparity measurement," *CVGIP: Image Understanding*, no. 53, pp. 198–210, 1991.

[21] J. Portilla and E. P. Simoncelli, "A parametric texture model based on joint statistics of complex wavelet coefficients," *Int'l J Computer Vision*, vol. 40, pp. 49–71, 2000.

[22] J. Daugman, "Statistical richness of visual phase information: update on recognizing persons by iris patterns," *Int'l J Computer Vision*, no. 45, pp. 25–38, 2001.

[23] E. P. Simoncelli, W. T. Freeman, E. H. Adelson, and D. J. Heeger, "Shiftable multi-scale transforms," *IEEE Trans Information Theory*, vol. 38, pp. 587–607, Mar 1992.

[24] D. A. Pollen and S. F. Ronner, "Phase relationships between adjacent simple cells in the cat," *Science*, no. 212, pp. 1409–1411, 1981.

[25] D. J. Heeger, "Half-squaring in responses of cat striate cells," *Visual Neuroscience*, no. 9, pp. 427–443, 1992.

[26] D. J. Heeger, "Normalization of cell responses in cat striate cortex," *Visual Neuroscience*, no. 9, pp. 181–197, 1992.

[27] O. Schwartz and E. P. Simoncelli, "Natural signal statistics and sensory gain control," *Nature Neuroscience*, no. 4, pp. 819–825, 2001.

[28] D. L. Ruderman and W. Bialek, "Seeing beyond the Nyquist limit," *Neural Comp.*, no. 4, pp. 682–690, 1992.

[29] G. Westheimer and S. P. McKee, "Spatial configurations for visual hyperacuity," *Vison Res.*, no. 17, pp. 941–947, 1977.

[30] W. S. Geisler, "Physical limits of acuity and hyperacuity," *J. Opti. Soc. America*, no. 1, pp. 775–782, 1984.
